# Graph Matching with Hierarchical Discrete Relaxation

**Richard C. Wilson and Edwin R. Hancock**
Department of Computer Science, University of York
York, Y01 5DD, UK.

## Abstract

Our aim in this paper is to develop a Bayesian framework for matching hierarchical relational models. The goal is to make discrete label assignments so as to optimise a global cost function that draws information concerning the consistency of match from different levels of the hierarchy. Our Bayesian development naturally distinguishes between intra-level and inter-level constraints. This allows the impact of reassigning a match to be assessed not only at its own (or peer) level of representation, but also upon its parents and children in the hierarchy.

## 1 Introduction

Hierarchical graphical structures are of critical importance in the interpretation of sensory or perceptual data. For instance, following the influential work of Marr [6] there has been sustained efforts at effectively organising and processing hierarchical information in vision systems. There are a plethora of concrete examples which include pyramidal hierarchies [3] that are concerned with multi-resolution information processing and conceptual hierarchies [4] which are concerned with processing at different levels of abstraction. Key to the development of techniques for hierarchical information processing is the desire to exploit not only the intra-level constraints applying at the individual levels of representation but also inter-level constraints operating between different levels of the hierarchy. If used effectively these inter-level constraints can be brought to bear on the interpretation of uncertain image entities in such a way as to improve the fidelity of interpretation achieved by single level means. Viewed as an additional information source, inter-level constraints can be used to resolve ambiguities that would persist if single-level constraints alone were used.

In the connectionist literature graphical structures have been widely used to represent probabilistic causation in hierarchical systems [5, 9]. Although this literature has provided a powerful battery of techniques, they have proved to be of limited use in practical sensory processing systems. The main reason for this is that the underpinning independence assumptions and the resulting restrictions on graph topology are rarely realised in practice. In particular there are severe technical problems in dealing with structures that contain loops or are not tree-like. One way to overcome this difficulty is to edit intractable structures to produce tractable ones [8].

Our aim in this paper is to extend this discrete relaxation framework to hierarchical graphical structures. We develop a label-error process to model the violation of both inter-level and intra-level constraints. These two sets of constraints have distinct probability distributions. Because we are concerned with directly comparing the topology graphical structures rather than propagating causation, the resulting framework is not restricted by the topology of the hierarchy. In particular, we illustrate the effectiveness of the method on amoral graphs used to represent scene-structure in an image interpretation problem. This is a heterogeneous structure [2, 4] in which different label types and different classes of constraint operate at different levels of abstraction. This is to be contrasted with the more familiar pyramidal hierarchy which is effectively homogeneous [1, 3]. Since we are dealing with discrete entities inter-level information communication is via a symbolic interpretation of the objects under consideration.

## 2    Hierarchical Consistency

The hierarchy consists of a number of levels, each containing objects which are fully described by their children at the level below. Formally each level is described by an attributed relational graph $G^l = (V^l, E^l, \mathbf{X}^l)$, $\forall l \in L$, with $L$ being the index-set of levels in the hierarchy; the indices $t$ and $b$ are used to denote the top and bottom levels of the hierarchy respectively. According to our notation for level $l$ of the hierarchy, $V^l$ is the set of nodes, $E^l$ is the set of intra-level edges and $\mathbf{X}^l = \{\underline{x}_u^l, \forall u \in V^l\}$ is a set of unary attributes residing on the nodes. The children or descendents which form the representation of an element $j$ at a lower level are denoted by $\mathcal{D}_j$. In other words, if $u^{l-1}$ is in $\mathcal{D}_j$ then there is a link in the hierarchy between element $j$ at level $l$ and element $u$ at level $l - 1$. According to our assumptions, the elements of $\mathcal{D}_j$ are drawn exclusively from $V^{l-1}$. The goal of performing relaxation operations is to find the match between scene graph $G_1$ and model graph $G_2$. At each individual level of the hierarchy this match is represented by a mapping function $f^l$, $\forall l \in L$, where $f^l : V_1^l \to V_2^l$.

The development of a hierarchical consistency measure proceeds along a similar line to the single-level work of Wilson and Hancock [10]. The quantity of interest is the MAP estimate for the mapping function $f$ given the available unary attributes, i.e. $f = \arg\max_{\hat{f}} P(\hat{f}^l, \forall l \in L | \mathbf{X}^l, \forall l \in L)$. We factorize the measurement information over the set of nodes by application of Bayes rule under the assumption of measurement independence on the nodes. As a result

$$P(f^l, \forall l \in L | \mathbf{X}^l, \forall l \in L) = \frac{1}{p(\mathbf{X}^l, \forall l \in L)} \left\{ \prod_{l \in L} \prod_{u \in V^l} p(\mathbf{x}_u^l | f^l(u)) \right\} P(f^l, \forall l \in L) \quad (1)$$

The critical modelling ingredient in developing a discrete relaxation scheme from the above MAP criterion is the joint prior for the mapping function, i.e. $P(f^l, \forall l \in L)$

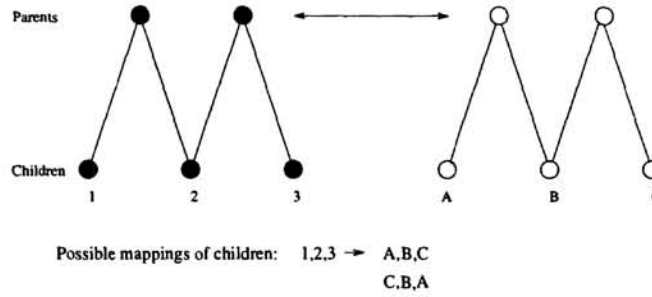

Possible mappings of children: 1,2,3 → A,B,C
C,B,A

Figure 1: Example constrained children mappings

which represents the influence of structural information on the matching process. The joint measurement density, $p(\mathbf{X}^l, \forall l \in L)$, on the other hand is a fixed property of the hierarchy and can be eliminated from further consideration.

Raw perceptual information resides on the lowest level of the hierarchy. Our task is to propagate this information upwards through the hierarchy. To commence our development, we assume that individual levels are conditionally dependent only on the immediately adjacent descendant and ancestor levels. This assumption allows the factorisation of the joint probability in a manner analogous to a Markov chain [3]. Since we wish to draw information from the bottom upwards, the factorisation commences from the highest level of labelling. The expression for the joint probability of the hierarchical labelling is

$$P(f^l, \forall l \in L) = P(f^b) \prod_{l \in L, l \neq t} P(f^{l+1} | f^l) \qquad (2)$$

We can now focus our attention on the conditional probabilities $P(f^{l+1} | f^l)$. These quantities express the probability of a labelling at the level $l+1$ given the previously defined labelling at the descendant level $l$. We develop tractable expressions for these probabilities by decomposing the hierarchical graph into convenient structural units. Here we build on the idea of decomposing single-level graphs into super-cliques that was successfully exploited in our previous work [10]. Super-cliques are the sets of nodes connected to a centre-object by intra-level edges. However, in the hierarchical case the relational units are more complex since we must also consider the graph-structure conveyed by inter-level edges.

We follow the philosophy adopted in the single-level case [10] by averaging the super-clique probabilities to estimate the conditional matching probabilities $P(f^{l+1} | f^l)$. If $\Gamma_j^l \subset f^l$ denotes the current match of the super-clique centred on the object $j \in V_1^l$ then we write

$$P(f^l | f^{l-1}) = \frac{1}{|V^l|} \sum_{j \in V^l} P(\Gamma_j^l | f^{l-1}) \qquad (3)$$

In order to model this probability, we require a dictionary of constraint relations for the corresponding graph sub-units (super-cliques) from the model graph $G_2$. The allowed mappings between the model graph and the data graph which preserve the topology of the graph structure at a particular level of representation are referred

to as "structure preserving mappings" or SPM's. It is important to note that we need only explore those mappings which are topologically identical to the super-clique centred on object $j$ and therefore the possible mappings of the child nodes are heavily constrained by the mappings of their parents (Figure 1). We denote the set of SPM's by $\mathcal{P}$. Since the set $\mathcal{P}$ is effectively the state-space of legal matching, we can apply the Bayes theorem to compute the conditional super-clique probability in the following manner

$$P(\Gamma_j^l | f^{l-1}) = \sum_{S \in \mathcal{P}} P(\Gamma_j^l | S, f^{l-1}) P(S | f^{l-1}) \qquad (4)$$

According to this expression, there are two distinct components to our model. The first involves the comparison between our mapped realisation of the super-clique from graph $G_1$, i.e. $\Gamma_j^l$, with the selected unit from graph $G_2$ and the mapping from level $l-1$. Here we take the view that once we have hypothesised a particular mapping $S$ from $\mathcal{P}$, the mapping $f^{l-1}$ provides us with no further information, i.e. $P(\Gamma_j^l | S, f^{l-1}) = P(\Gamma_j^l | S)$. The matched super-clique $\Gamma_j^l$ is conditionally independent given a mapping from the set of SPM's and we may write the first term as $P(\Gamma_j^l | S)$. In other words, this first conditional probability models only intra-level constraints. The second term is the significant one in evaluating the impact inter-level constraints on the labelling at the previous level. In this term the probability of the hypothesised mapping $S$ is conditioned according to the match of the child level $l$.

All that remains now is to evaluate the conditional probabilities. Under the assumption of memoryless matching errors, the first term may be factorised over the marginal probabilities for the assigned matches $\gamma_i^l$ on the individual nodes of the matched super-clique $\Gamma_j^l$ given their counterparts $s_i$ belonging to the structure preserving mapping $S$. In other words,

$$P(\Gamma_j^l | S) = \prod_{\gamma_i^l \in \Gamma_j^l} P(\gamma_i^l | s_i) \qquad (5)$$

In order to proceed we need to specify a probability distribution for the different matching possibilities. There are three cases. Firstly, the match $\gamma_i^l$ may be to a dummy-node $d$ inserted into $\Gamma_j^l$ to raise it to the same size as $S$ so as to facilitate comparison. This process effectively models structural errors in the data-graph. The second and third cases, relate to whether the match is correct or in error. Assuming that dummy node insertions may be made with probability $P_s$ and that matching errors occur with probability $P_e$, then we can write down the following distribution rule

$$P(\gamma_i^l | s_i) = \begin{cases} P_s & \text{if } \gamma_i^l = d \text{ or } s_i = d \\ (1 - P_e)(1 - P_s) & \text{if } \gamma_i^l = s_i \\ P_e(1 - P_s) & \text{otherwise} \end{cases} \qquad (6)$$

The second term in Equation (5) is more subtle; it represents the conditional probability of the SPM $S$ given a previously determined labelling at the level below. However, the mapping contains labels only from the current level $l$, not labels from level $l-1$. We can reconcile this difference by noting that selection of a particular mapping at level $l$ limits the number of consistent mappings allowed topologically at the level below. In other words if one node is mapped to another at level $l$,

the consistent interpretation is that the children of the nodes must match to each other. Provided that a set of mappings is available for the child-nodes, then this constraint can be used to model $P(S|f^{l-1})$. The required child-node mappings are referred to as "Hierarchy Preserving Mappings" or HPM's. It is these hierarchical mappings that lift the requirements for moralization in our matching scheme, since they effectively encode potentially incestuous vertical relations. We will denote the set of HPM's for the descendants of the SPM $S$ as $Q_S$ and a member of this set as $Q = \{q_i, \forall i \in \mathcal{D}_j\}$. Using this model the conditional probability $P(S|f^{l-1})$ is given by

$$P(S|f^{l-1}) = \sum_{Q \in Q_s} P(S|Q, f^{l-1})P(Q|f^{l-1}) \tag{7}$$

Following our modelling of the intra-level probabilities, in this inter-level case assume that $S$ is conditionally independent of $f^{l-1}$ given $Q$, i.e. $P(S|Q, f^{l-1}) = P(S|Q)$.

Traditionally, dictionary based hierarchical schemes have operated by using a labelling determined at a preceding level to prune the dictionary set by elimination of vertically inconsistent items [4]. This approach can easily be incorporated into our scheme by setting $P(Q|f^{l-1})$ equal to unity for consistent items and to zero for those which are inconsistent. However we propose a different approach; by adopting the same kind of label distribution used in Equation 6 we can grade the SPM's according to their consistency with the match at level $l-1$, i.e. $f^{l-1}$. The model is developed by factorising over the child nodes $q_i \in Q$ in the following manner

$$P(Q|f^{l-1}) = \prod_{q_i \in Q} P(q_i|\gamma_i^{l-1}) \tag{8}$$

The conditional probabilities are assigned by a re-application of the distribution rule given in Equation (6), i.e.

$$P(q_i|f^{l-1}) = \begin{cases} P_s & \text{if dummy node match} \\ (1 - P_e)(1 - P_s) & \text{if } q_i = \gamma_i^{l-1} \\ P_e(1 - P_s) & \text{otherwise} \end{cases} \tag{9}$$

For the conditional probability of the SPM given the HPM $Q$, we adopt a simple uniform model under the assumption that all legitimate mappings are equivalent, i.e. $P(S|Q) = P(S) = \frac{1}{|\mathcal{P}|}$.

The various simplifications can be assembled along the lines outlined in [10] to develop a discrete update rule for matching the two hierarchical structures. The MAP update decision depends only on the label configurations residing on levels $l-1$, $l$ and $l+1$ together with the measurements residing on level $l$. Specifically, the level $l$ matching configuration satisfies the condition

$$f^l = \arg\max_{\hat{f}^l} \left\{ \prod_{j \in V_1^l} p(\underline{x}_j^l|\hat{f}^l(j)) \right\} P(f^{l-1}|\hat{f}^l)P(\hat{f}^l|f^{l+1}) \tag{10}$$

Here consistency of match between levels $l$ and $l-l$ of the hierarchy is gauged by

the quantity

$$P(f^{l-1}|f^l) = \frac{1}{V_1^l} \sum_{i \in V_1^l} \sum_{S \in \mathcal{P}} \frac{K(\Gamma_i^l)}{Q_S} \quad \exp \quad \left[-(k_e H(\Gamma_i^l, S) + k_s \Phi(\Gamma_i^l, S))\right]$$

$$\sum_{Q \in \mathcal{Q}_S} K(\Gamma_i^{l-1}) \quad \exp \quad \left[-(k_e H(\Gamma_i^{l-1}, Q) + k_s \Phi(\Gamma_i^{l-1}, Q))\right] \quad (11)$$

In the above expression $H(\Gamma_j, S)$ is the "Hamming distance" which counts the number of label conflicts between the assigned match $\Gamma_j$ and the structure preserving mapping $S$. This quantity measures the consistency of the matched labels. The number of dummy nodes inserted into $\Gamma_j$ by the mapping $S$ is denoted by $\Phi(\Gamma_j, S)$. This second quantity measures the structural compatibility of the two hierarchical graphs. The exponential constants $k_e = \ln \frac{(1-P_e)(1-P_s)}{P_e}$ and $k_s = \ln \frac{1-P_s}{P_s}$ are related to the probabilities of structural errors and mis-assignment errors. Finally, $K(\Gamma_j) = (1 - P_e)(1 - P_e)^{|\Gamma_j|}$ is a normalisation constant. Finally, it is worth pointing out that the discrete relaxation scheme of Equation (10) can be applied at any level in the hierarchy. In other words the process can be operated in top-down or bottom-up modes if required.

## 3  Matching SAR Data

In our experimental evaluation of the discrete relaxation scheme we will focus on the matching of perceptual groupings of line-segments in radar images. Here the model graph is elicited from a digital map for the same area as the radar image. The line tokens extracted from the radar data correspond to hedges in the landscape. These are mapped as quadrilateral field boundaries in the cartographic model. To support this application, we develop a hierarchical matching scheme based on line-segments and corner groupings. The method used to extract these features from the radar images is explained in detail in [10]. Straight line segments extracted from intensity ridges are organised into corner groupings. The intra-level graph is a constrained Delaunay triangulation of the line-segments. Inter-level relations represent the subsumption of the bottom-level line segments into corners.

The raw image data used in this study is shown in Figure 2a. The extracted line-segments are shown in Figure 2c. The map used for matching is shown in Figure 2b. The experimental matching study is based on 95 linear segments in the SAR data and 30 segments contained in the map. However only 23 of the SAR segments have feasible matches within the map representation. Figure 2c shows the matches obtained by non-hierarchical means. The lines are coded as follows; the black lines are correct matches while the grey lines are matching errors. With the same coding scheme Figure 2d. shows the result obtained using the hierarchical method outlined in this paper. Comparing Figures 2c and 2d it is clear that the hierarchical method has been effective at grouping significant line structure and excluding clutter. To give some idea of relative performance merit, in the case of the non-hierarchical method, 20 of the 23 matchable segments are correctly identified with 75 incorrect matches. Application of the hierarchical method gives 19 correct matches, only 17 residual clutter segments with 59 nodes correctly labelled as clutter.

## 4  Conclusions

We have developed graph matching technique which is tailored to hierarchical relational descriptions. The key element is this development is to quantify the match-

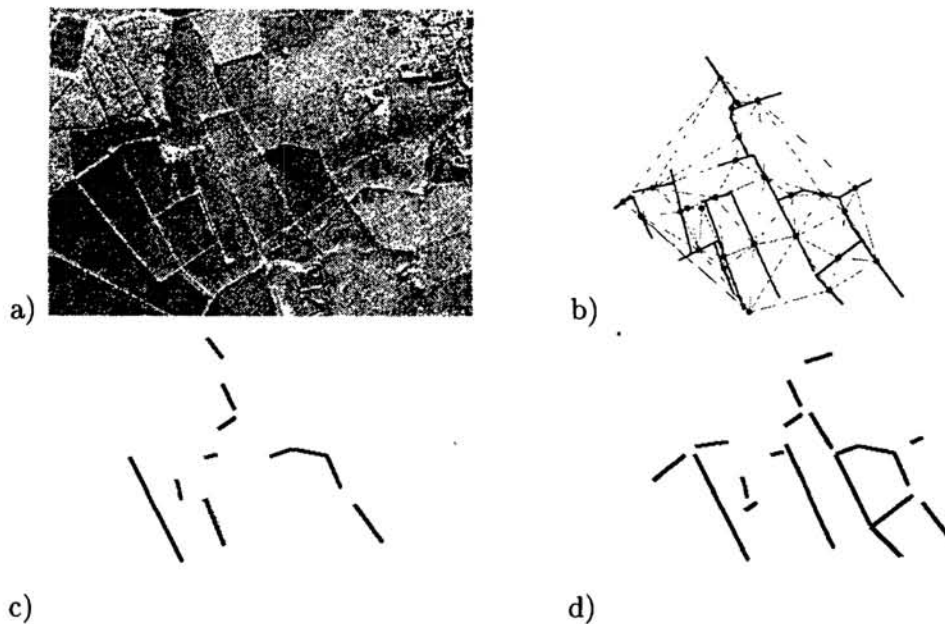

Figure 2: Graph editing: a) Original image, b) Digital map, c) Non hierarchical match , d) Hierarchical match.

ing consistency using the concept of hierarchy preserving mappings between two graphs. Central to the development of this novel technique is the idea of computing the probability of a particular node match by drawing on the topologically allowed mappings of the child nodes in the hierarchy. Results on image data with lines and corners as graph nodes reveal that the technique is capable of matching perceptual groupings under moderate levels of corruption.

# References

[1] F. Cohen and D. Cooper. *Simple Parallel Hierarchical and Relaxation Algorithms for Segmenting Non-Causal Markovian Random Fields.* IEEE PAMI, **9**, 1987, pp.195–219.

[2] L. Davis and T. Henderson. *Hierarchical Constraint Processes for Shape Analysis.* IEEE PAMI, **3**, 1981, pp.265–277.

[3] B. Gidas. *A Renormalization Group Approach to Image Processing Problems.* IEEE PAMI, **11**, 1989, pp.164–180.

[4] T. Henderson. *Discrete Relaxation Techniques.* Oxford University Press, 1990.

[5] D.J. Spiegelhalter and S.L. Lauritzen, *Sequential updating of conditional probabilities on directed Graphical structures*, Networks, 1990, **20**, pp.579-605.

[6] D. Marr, *Vision*. W.H. Freeman and Co., San Francisco.

[7] J. Pearl, *Probabilistic Reasoning in Intelligent Systems*, Morgan Kaufmann, 1988.

[8] M. Meila and M. Jordan, *Optimal triangulation with continuous cost functions*, Advances in Neural Information Processing Systems 9, to appear 1997.

[9] P.Smyth, D. Heckerman, M.I. Jordan, *Probabilistic independence networks for hidden Markov probability models*, Neural Computation, **9**, 1997, pp. 227-269.

[10] R.C. Wilson and E. R. Hancock, *Structural Matching by Discrete Relaxation.* IEEE PAMI, **19**, 1997, pp.634–648.

IEEE PAMI, June 1997.
